# Gaussian sampling by local perturbations

**George Papandreou**
Department of Statistics
University of California, Los Angeles
gpapan@stat.ucla.edu

**Alan L. Yuille**
Depts. of Statistics, Computer Science & Psychology
University of California, Los Angeles
yuille@stat.ucla.edu

## Abstract

We present a technique for exact simulation of Gaussian Markov random fields (GMRFs), which can be interpreted as locally injecting noise to each Gaussian factor independently, followed by computing the mean/mode of the perturbed GMRF. Coupled with standard iterative techniques for the solution of symmetric positive definite systems, this yields a very efficient sampling algorithm with essentially linear complexity in terms of speed and memory requirements, well suited to extremely large scale probabilistic models. Apart from synthesizing data under a Gaussian model, the proposed technique directly leads to an efficient unbiased estimator of marginal variances. Beyond Gaussian models, the proposed algorithm is also very useful for handling highly non-Gaussian continuously-valued MRFs such as those arising in statistical image modeling or in the first layer of deep belief networks describing real-valued data, where the non-quadratic potentials coupling different sites can be represented as finite or infinite mixtures of Gaussians with the help of local or distributed latent mixture assignment variables. The Bayesian treatment of such models most naturally involves a block Gibbs sampler which alternately draws samples of the conditionally independent latent mixture assignments and the conditionally multivariate Gaussian continuous vector and we show that it can directly benefit from the proposed methods.

## 1 Introduction

Using Markov random fields (MRFs) one can capture global statistical properties in large scale probabilistic networks while only explicitly modeling the interactions of neighboring sites. First introduced in statistical physics, MRFs and related models such as Boltzmann machines have proved particularly successful in computer vision and machile learning tasks such as image segmentation, signal recovery, texture modeling, classification, and unsupervised learning [1, 3, 5]. Drawing random samples from MRFs and juxtaposing them with real data allows one to directly assess the model quality. Sampling of MRFs also plays an important role within algorithms for model parameter fitting [7], signal estimation, and in image analysis for texture synthesis or inpainting [16, 19, 37]. The simplest but typically very slow way to draw random samples from MRFs is through single-site Gibbs sampling, a Markov chain Monte-Carlo (MCMC) algorithm in which one visits each node in the network and stochastically updates its state given the states of its neighbors [5].

Gaussian Markov random fields (GMRFs) are an important MRF class describing continuous variables linked by quadratic potentials [3,22,29,33] – see Sec. 2. They are very useful both for modeling inherently Gaussian data and as building blocks for constructing more complex models. In this paper we study a technique which allows drawing exact samples from a GMRF in a single shot by first perturbing it and then computing the least energy configuration of the perturbed model. The perturbation involved amounts to independently injecting noise to each of the Gaussian factors/potentials in a fully distributed manner, as discussed in detail in Sec. 3. This reduction of sampling to quadratic energy minimization allows us to employ as black-box GMRF simulator any existing algorithm for MAP computation which is effective for a particular Gaussian graphical model.

The reliability of the most likely solution in a Gaussian model is characterized by the marginal variances. Marginal variances also arise in computations within non-linear sparse Bayesian learning and compressed sensing models [11, 26, 32]. However, their computation can be very challenging and a host of sophisticated techniques have been developed for this purpose, which often only apply to restricted classes of models [12, 24, 25, 28]. Being able to efficiently sample from a GMRF makes it practical to employ the generic sample-based estimator for computing Gaussian variances, as discussed in Sec. 4. This estimator, whose accuracy is independent of the problem size, is particularly attractive if only relatively rough variance estimates suffice, as is often the case in practice.

Gaussian models have proven inadequate for image modeling as they fail to capture important aspects of natural image statistics such as the heavy tails in marginal histograms of linear filter responses. Nevertheless, much richer statistical image tools can be built if we also incorporate into our models latent variables or allow nonlinear interactions between multiple Gaussian fields and thus the GMRF sampling technique we describe here is very useful within this wider setting [10, 16, 19, 34]. In Sec. 5 we discuss the integration of our GMRF sampling algorithm in a block-Gibbs sampling context, where the conditionally Gaussian continuous variables and the conditionally independent latent variables are sampled alternately. The most straightforward way to capture the heavy tailed histograms of natural images is to model each filter response with a Gaussian mixture expert, thus using a single discrete assignment variable at each factor [16, 23]. However, our efficient GMRF algorithm can also be used in conjunction with Gaussian scale mixture (GSM) models for which the latent scale variable is continuous [2]; we demonstrate this in the context of Bayesian signal restoration by sampling from the posterior distribution under a total variation (TV) prior, employing the GSM characterization of the Laplacian density. Further, our sampling technique also applies when the latent variables are distributed, with each hidden variable affecting multiple experts. An interesting case we examine is the recently proposed factored Gaussian restricted Boltzmann machine (GRBM) of [18], which takes into account residual correlations among visible units by modeling them as a multivariate GMRF, conditional on the distributed state of an adjacent layer of discrete hidden units. We show that we can effectively replace the hybrid Monte Carlo sampler used by [18] with a block-Gibbs sampler in which the visible conditionally Gaussian units are sampled collectively by local perturbations, potentially allowing extension of the current patch-based model to a full-image factored GRBM, as has been recently done for the fields of independent experts model [19, 23].

Our GMRF sampling algorithm relies on a property of Gaussian densities (see Sec. 3) which, in a somewhat different form, has appeared before in the statistics literature [21, 22]. However, [21, 22] emphasize direct matrix factorization methods for solving the linear system arising in computing the Gaussian mean, which cannot handle the large models we consider here and do not discuss models with hidden variables. Variations of the sampling technique we study here have been also used in the image modeling work of [16] and very recently of [23]. However the sampling technique in these papers is used as a tool and not studied by itself. Apart from highlighting the power and versatility of the efficient GMRF sampling algorithm and drawing the machine learning community's attention to it, our main novel contributions in this paper are: (1) Our interpretation of the Gaussian sampling algorithm as local factor perturbation followed by mode computation, which highlights its distributed nature and implies that any Gaussian mean computation routine can be equally effectively employed for GMRF sampling; (2) the application of the efficient sampling algorithm in rapid sampling and variance estimation of very large Gaussian models; and (3) the demonstration that, in the presence of hidden variables, it can be effectively integrated in a block-Gibbs sampler not only in discrete but also in continuous GSM models and in conjunction not only with local but also with distributed latent assignment representations.

## 2 Gaussian graphical models

### 2.1 The linear Gaussian model

We are working in the context of linear Gaussian models [20], in which a hidden vector $\mathbf{x} \in \mathbb{R}^N$ is assumed to follow a prior distribution $P(\mathbf{x})$ and noisy linear measurements $\mathbf{y} \in \mathbb{R}^M$ of it are drawn with likelihood $P(\mathbf{y}|\mathbf{x})$. Specifically:

$$P(\mathbf{x}) \propto \mathcal{N}(\mathbf{Gx}; \boldsymbol{\mu_p}, \boldsymbol{\Sigma_p}) \propto \exp\left(-\tfrac{1}{2}\mathbf{x}^T \mathbf{J_x x} + \mathbf{k_x}^T \mathbf{x}\right)$$

$$P(\mathbf{y}|\mathbf{x}) = \mathcal{N}(\mathbf{y}; \mathbf{Hx} + \mathbf{c}, \boldsymbol{\Sigma_n}) \propto \exp\left(-\tfrac{1}{2}\mathbf{x}^T \mathbf{J_{y|x} x} + \mathbf{k_{y|x}}^T \mathbf{x} - \tfrac{1}{2}\mathbf{y}^T \boldsymbol{\Sigma_n}^{-1} \mathbf{y}\right) \quad (1)$$

where $\mathcal{N}(\mathbf{x}; \boldsymbol{\mu}, \boldsymbol{\Sigma}) = |2\pi\boldsymbol{\Sigma}|^{-1/2} \exp\left(-\frac{1}{2}(\mathbf{x} - \boldsymbol{\mu})^T \boldsymbol{\Sigma}^{-1}(\mathbf{x} - \boldsymbol{\mu})\right)$ denotes the multivariate Gaussian density on $\mathbf{x}$ with mean $\boldsymbol{\mu}$ and covariance $\boldsymbol{\Sigma}$. It is convenient to express the prior and likelihood Gaussian densities on $\mathbf{x}$ in Eq. (1) in information form; the respective parameters are

$$\mathbf{J_x} = \mathbf{G}^T \boldsymbol{\Sigma_p}^{-1} \mathbf{G}, \;\; \mathbf{k_x} = \mathbf{G}^T \boldsymbol{\Sigma_p}^{-1} \boldsymbol{\mu_p} \quad \text{and} \quad \mathbf{J_{y|x}} = \mathbf{H}^T \boldsymbol{\Sigma_n}^{-1} \mathbf{H}, \;\; \mathbf{k_{y|x}} = \mathbf{H}^T \boldsymbol{\Sigma_n}^{-1}(\mathbf{y} - \mathbf{c}) \,. \tag{2}$$

We recall that the information form of the Gaussian density $\mathcal{N}_I(\mathbf{x}; \mathbf{k}, \mathbf{J}) \propto \exp\left(-\frac{1}{2}\mathbf{x}^T \mathbf{J}\mathbf{x} + \mathbf{k}^T \mathbf{x}\right)$ employs the precision matrix $\mathbf{J}$ and the potential vector $\mathbf{k}$ [13]. If $\mathbf{J}$ is invertible, then the standard and information representations are equivalent, with $\boldsymbol{\mu} = \mathbf{J}^{-1}\mathbf{k}$ and $\boldsymbol{\Sigma} = \mathbf{J}^{-1}$, but the information form with $\mathbf{J}$ symmetric positive semidefinite is also convenient for describing degenerate Gaussian densities. Further, the precision matrix directly reveals dependencies between subsets of variables in the network: $x_i$ and $x_j$ are conditionally independent, given the values of the remaining components of $\mathbf{x}$, iff $J_{i,j} = 0$, while, in general, $\Sigma_{i,j} \neq 0$; this implies that $\mathbf{J}$ is typically much sparser than $\boldsymbol{\Sigma}$ for GMRF models, as further discussed in Sec. 2.2.

By Bayes' rule the posterior distribution of $\mathbf{x}$ given $\mathbf{y}$ is the product of the prior and likelihood terms and also has Gaussian density

$$P(\mathbf{x}|\mathbf{y}) = \mathcal{N}(\mathbf{x}; \boldsymbol{\mu}, \boldsymbol{\Sigma}), \quad \text{with}$$
$$\boldsymbol{\mu} = \mathbf{J}^{-1}\left(\mathbf{G}^T \boldsymbol{\Sigma_p}^{-1} \boldsymbol{\mu_p} + \mathbf{H}^T \boldsymbol{\Sigma_n}^{-1}(\mathbf{y} - \mathbf{c})\right) \;\; \text{and} \;\; \boldsymbol{\Sigma}^{-1} = \mathbf{J} = \mathbf{G}^T \boldsymbol{\Sigma_p}^{-1} \mathbf{G} + \mathbf{H}^T \boldsymbol{\Sigma_n}^{-1} \mathbf{H} \,. \tag{3}$$

We assume $\mathbf{J} = \mathbf{J_x} + \mathbf{J_{y|x}}$ to be invertible, although we allow for singular $\mathbf{J_x}$ and/or $\mathbf{J_{y|x}}$; in other words, the prior and likelihood jointly define a normalizable Gaussian density on $\mathbf{x}$, although each of them on its own may leave a subspace of $\mathbf{x}$ unconstrained.

## 2.2 Gaussian Markov random fields

The $K$ rows of $\mathbf{G} = [\mathbf{g}_1^T; \ldots; \mathbf{g}_K^T]$ and the $M$ rows of $\mathbf{H} = [\mathbf{h}_1^T; \ldots; \mathbf{h}_M^T]$ can be seen as two sets of length-$N$ linear filters. The respective filter responses $\mathbf{Gx}$ and $\mathbf{Hx}$ determine the prior and likelihood models of Eq. (1). We define the filter set $\mathbf{F} = [\mathbf{f}_1^T; \ldots; \mathbf{f}_L^T]$, $L = K+M$, as the union of $\{\mathbf{g}_k\}$ and $\{\mathbf{h}_m\}$ and further assume that any two filter responses are conditionally independent given $\mathbf{x}$ or, equivalently, that the covariance matrices in Eq. (1) are diagonal, $\boldsymbol{\Sigma_p} = \mathrm{diag}(\Sigma_{\mathbf{p},1}, \ldots, \Sigma_{\mathbf{p},K})$ and $\boldsymbol{\Sigma_n} = \mathrm{diag}(\Sigma_{\mathbf{n},1}, \ldots, \Sigma_{\mathbf{n},M})$. Also let $\boldsymbol{\mu_p} = (\mu_{\mathbf{p},1}; \ldots; \mu_{\mathbf{p},K})$, $\mathbf{y} = (y_1; \ldots; y_M)$, and $\mathbf{c} = (c_1; \ldots; c_M)$. Then the posterior factorizes as a product of $L$ Gaussian experts

$$P(\mathbf{x}|\mathbf{y}) \propto \prod_{l=1}^{L} \exp\left(-\frac{1}{2}\mathbf{x}^T \mathbf{J}_l \mathbf{x} + \mathbf{k}_l^T \mathbf{x}\right) \propto \prod_{l=1}^{L} \mathcal{N}(\mathbf{f}_l^T \mathbf{x}; \mu_l, \Sigma_l), \tag{4}$$

where the variances are $\Sigma_l = \Sigma_{\mathbf{p},l}$, $l = 1 \ldots K$, for the factors that come from the prior term and $\Sigma_l = \Sigma_{\mathbf{n},l-K}$, $l = K+1 \ldots K+M$, for those that come from the likelihood term; the corresponding means are $\mu_l = \mu_{\mathbf{p},l}$ and $\mu_l = y_{l-K} - c_{l-K}$, respectively. Comparing with Eq. (3), we see that the posterior Gaussian information parameters split additively as $\mathbf{J} = \sum_{l=1}^{L} \mathbf{J}_l$ and $\mathbf{k} = \sum_{l=1}^{L} \mathbf{k}_l$. The individual Gaussian factors have potential vectors $\mathbf{k}_l = \mathbf{f}_l \Sigma_l^{-1} \mu_l$ and rank-one precision matrices $\mathbf{J}_l = \mathbf{f}_l \Sigma_l^{-1} \mathbf{f}_l^T$. Since $\mathbf{J}$ is invertible, $L \geq N$. We see that there is a one-to-one correspondence between factors and filters; moreover, the $(i,j)$ entry of $\mathbf{J}_l$ is non-zero iff both $i$ and $j$ entries of $\mathbf{f}_l$ are non-zero. If the filter has $T_l$ non-zero elements, then the corresponding Gaussian factor will couple the $T_l$ variables in the clique $\mathbf{x}_{[l]}$. The resulting GMRF is depicted in a factor graph form in Fig. 1(a). It is straightforward to jointly model conditionally dependent filter responses by letting $\boldsymbol{\Sigma_p}$ or $\boldsymbol{\Sigma_n}$ have block diagonal structure, yielding multivariate Gaussian factors in Eq. (4).

## 2.3 Inference: Efficiently computing the posterior mean

Conceptually, the Gaussian posterior distribution is fully characterized by the posterior mean $\boldsymbol{\mu}$ and covariance matrix $\boldsymbol{\Sigma}$, which are given in closed form in Eq. (3): $\boldsymbol{\mu}$ is the solution of a set of linear equations whose system matrix is the $N \times N$ precision matrix $\mathbf{J}$, while $\boldsymbol{\Sigma} = \mathbf{J}^{-1}$. However, naively computing these quantities can be prohibitively expensive when working with high-dimensional models, requiring $\mathcal{O}(N^3)$ computation and $\mathcal{O}(N^2)$ space. For example, a typical 1 MP image model involves $N = 10^6$ variables; the corresponding symmetric covariance matrix $\boldsymbol{\Sigma}$ is generally dense and occupies as much space as about $5 \times 10^5$ equally-sized images.

Thankfully, for the GMRF models mostly used in practice, there exist powerful inference algorithms which avoid explicitly inverting the system matrix $\mathbf{J}$. In certain special cases direct methods

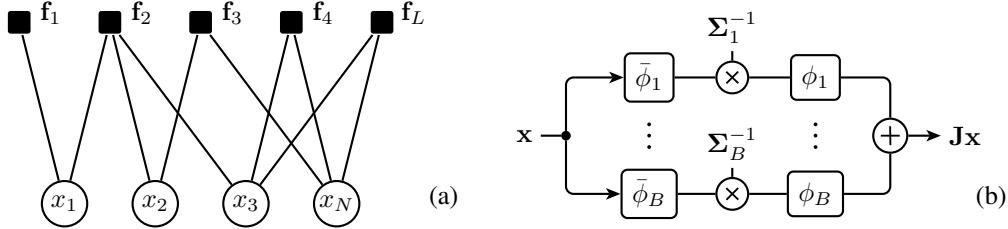

Figure 1: (a) The factor graph for the posterior GMRF contains the union $\mathbf{f}_{1:L}$ of prior and likelihood factors/filters. An edge between a filter and a site means that the corresponding coefficient is non-zero. The variables connected to each factor comprise a clique of the GMRF. (b) Filterbank implementation of matrix-vector multiplication $\mathbf{Jx}$ arising in CG ($\bar{\phi}$ is the spatial mirror of $\phi$).

are applicable for computing the mode, marginal variances, and samples from the posterior. For example, spatially homogeneous GMRFs give rise to a block-circulant precision matrix and exact computations can be carried out in $\mathcal{O}(N \log N)$ complexity with DFT-based techniques [10]. Exact inference can also be carried out in chain or tree-structured GMRFs using $\mathcal{O}(N)$ Kalman filter equations which correspond to belief propagation (BP) updates recursively in time or scale [36]. A related direct approach which in the context of GMRFs has been studied in detail by [21, 22] relies on the Cholesky factorization of the precision matrix by efficient sparse matrix techniques, which typically re-order the variables in $\mathbf{x}$ so as to minimize the bandwidth $W$ of $\mathbf{J}$. The resulting algorithm has $\mathcal{O}(W^2 N)$ speed and $\mathcal{O}(WN)$ space complexity, which is still quite expensive for very large scale 2-D lattice image models, since the bandwidth $W$ increases linearly with the spatial extent of the image and the support of the filters.

More generally, for large scale and arbitrarily structured GMRFs one needs to resort to iterative techniques such as conjugate gradients, multigrid, or loopy BP in order to approximately solve the linear system in Eq. (3) and recover the most likely solution $\boldsymbol{\mu}$. Conjugate gradients (CG) [6] are generally applicable in our setup since the system matrix is positive definite. Each CG iteration involves a single matrix-vector multiplication $\mathbf{Jx}$. By Sec. 2.2, this essentially amounts to computing the filter responses $z_l = \mathbf{f}_l^T \mathbf{x}$ and the backprojection $\sum_{l=1}^L \Sigma_l^{-1} z_l \mathbf{f}_l$, which respectively involves sending messages from the variables to the factors and back in the diagram of Fig. 1(a). The GMRFs arising in image modeling are typically defined on the image responses to a bank of linear filters $\{\phi_\ell\}, \ell = 1 \ldots, B$; the spatial translation of each filter kernel $\phi_\ell$ induces a subset of factors. In this context, the matrix-vector multiplication $\mathbf{Jx}$ in CG corresponds to convolutions and element-wise multiplications, as shown in the filterbank diagram of Fig. 1(b). The time complexity per iteration is thus low, typically $\mathcal{O}(N)$ or $\mathcal{O}(N \log N)$, provided that the filter kernels $\phi_\ell$ have small spatial support or correspond to wavelet or Fourier atoms for which fast discrete transforms exist, while computations can also be carried out in the GPU. The memory overhead is also minimal, $\mathcal{O}(N)$, as CG employs only 3 or 4 auxiliary length-$N$ vectors. The convergence rate of CG is largely problem-dependent, but in many cases a relatively small number of iterations suffice to bring us close enough to the solution, especially if an effective preconditioner is used [6]. Multigrid algorithms also apply in certain of the GMRF models we consider, especially those related to physics-based variational energy and PDE formulations [29, 31]. When multigrid applies, as in the example of Sec. 3, it recovers the solution after a fixed number of iterations (independent of the problem size) and has optimal $\mathcal{O}(N)$ time and space complexity. Loopy BP is a powerful distributed iterative method for computing $\boldsymbol{\mu}$ which is guaranteed to converge for certain GMRF classes [13, 33].

## 3   Gaussian sampling by independent factor perturbations

Unlike direct methods, the iterative techniques discussed in Sec. 2.3 have been typically restricted to computing the posterior mode $\boldsymbol{\mu}$ and considered less suited to posterior sampling or variance computation (but see Sec. 4). However, as the following result shows, exact sampling from a linear Gaussian model can be reduced to computing the mode of a Gaussian model with identical precision matrix $\mathbf{J}$ but randomly perturbed potential vector $\tilde{\mathbf{k}}$, and thus the powerful iterative methods for recovering the mean can be used unmodified for sampling in large scale GMRFs. Specifically:

**Algorithm.** *A sample* $\mathbf{x}_s$ *from the posterior distribution* $P(\mathbf{x}|\mathbf{y}) = \mathcal{N}(\mathbf{x}; \boldsymbol{\mu}, \boldsymbol{\Sigma})$ *of Eq.* (3) *can be drawn using the following procedure: (1) Perturb the prior mean filter responses* $\tilde{\boldsymbol{\mu}}_{\mathbf{p}} \sim \mathcal{N}(\boldsymbol{\mu}_{\mathbf{p}}, \boldsymbol{\Sigma}_{\mathbf{p}})$.

*(2) Perturb the measurements $\tilde{\mathbf{y}} \sim \mathcal{N}(\mathbf{y}, \mathbf{\Sigma_n})$. (3) Use the procedure for computing the posterior mode keeping the same system matrix $\mathbf{J}$, only replacing $\boldsymbol{\mu_p}$ and $\mathbf{y}$ with their perturbed versions:* $\mathbf{x}_s = \mathbf{J}^{-1} \left( \mathbf{G}^T \mathbf{\Sigma_p}^{-1} \tilde{\boldsymbol{\mu}}_{\mathbf{p}} + \mathbf{H}^T \mathbf{\Sigma_n}^{-1} (\tilde{\mathbf{y}} - \mathbf{c}) \right)$.

Indeed, $\mathbf{x}_s$ is a Gaussian random vector, as linear combination of Gaussians, and has the desired mean $E\{\mathbf{x}_s\} = \boldsymbol{\mu}$ and covariance $E\{(\mathbf{x}_s - \boldsymbol{\mu})(\mathbf{x}_s - \boldsymbol{\mu})^T\} = \mathbf{J}^{-1} = \mathbf{\Sigma}$, as can readily be verified. Clearly, solving the corresponding linear system approximately will only yield an approximate sample. The reduction above implies that posterior sampling under the linear Gaussian model is computationally as hard as mode computation, provided that the structure of $\mathbf{\Sigma_p}$ and $\mathbf{\Sigma_n}$ allows efficient sampling from the corresponding distributions, using, e.g., the direct methods of Sec. 2.3. This algorithm is central to our paper; variations of it have appeared previously [16, 22, 23].

The sampling algorithm takes a particularly simple and intuitive form for the GMRFs discussed in Sec. 2.2. In this case $\mathbf{\Sigma_p}$ and $\mathbf{\Sigma_n}$ are diagonal and thus for sampling we perturb independently the factor means $\tilde{\mu}_l \sim \mathcal{N}(\mu_l, \Sigma_l)$, $l = 1 \ldots L$, followed by finding the mode of the so perturbed GMRF in Eq. (4). The perturbation can be equivalently seen in the information parameterization as injecting Gaussian noise to each potential vector by $\tilde{\mathbf{k}}_l = \mathbf{k}_l + \mathbf{f}_l \Sigma_l^{-1/2} \epsilon_l$, with $\epsilon_l \sim \mathcal{N}(0, 1)$, a simple local operation carried out independently at each factor of the diagram in Fig. 1(a).

To demonstrate the power of this algorithm, we show in Fig. 2 an image inpainting example in which we fill in the occluded parts of an $498 \times 495$ image under a 2-D thin-membrane prior GMRF model [12, 29, 31], in which the Gaussian factors are induced by the first-order spatial derivative filters $\phi_1 = \begin{bmatrix} -1 & 1 \end{bmatrix}$ and $\phi_2 = \begin{bmatrix} -1 & 1 \end{bmatrix}^T$. The shared variance parameter $\Sigma_l$ for the experts has been matched to the variance of the image derivative histogram. The presence of randomly placed measurements makes the problem non-stationary and thus Fourier domain techniques are not applicable. Finding the posterior mean of this model amounts to solving a quadratic energy minimization problem in which the non-occluded pixels are clamped to their observed values and corresponds to a Laplace PDE problem with non-homogeneous regularization, which can be tackled very efficiently with multigrid techniques [31]. To transform this efficient MAP computation technique into a powerful sampling algorithm for the thin-membrane GMRF, it suffices to inject noise to the factors, only perturbing the linear system's right hand side. Using a multigrid solver originally developed for solving PDE problems, we can draw about 4 posterior samples per second from the 2-D thin-membrane model of Fig. 2, which is particularly impressive given its size; the multilevel Gibbs sampling technique of [30] is the only other algorithm that could potentially achieve such speed in a similar setup, yet it cannot produce exact single-shot samples as our algorithm can.

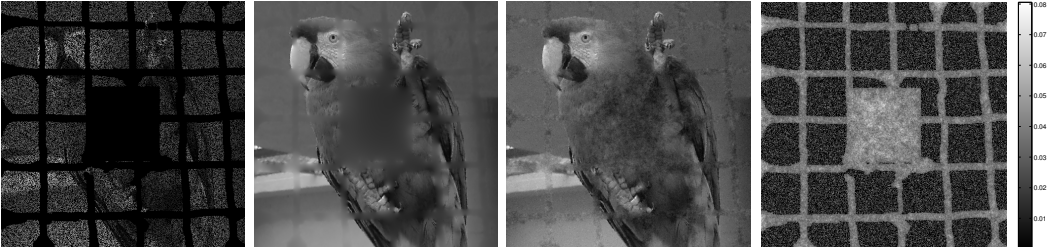

Figure 2: Image inpainting by exact sampling from the posterior under a 2-D thin-membrane prior GMRF model, conditional on the image values at the known sites. From left to right, the masked image (big occluded areas plus 50% missing pixels), the posterior mean, a posterior sample obtained by our perturbed GMRF sampling algorithm, and the sample-based estimate of the posterior standard deviation (square root of the variance) using 20 samples (image values are between 0 and 1).

## 4   Posterior variance estimation

It is often desirable not only to compute the mode $\boldsymbol{\mu}$ but also recover aspects of the covariance structure in the posterior distribution. As we have discussed in Sec. 2.1, for very large models the fully-dense covariance matrix $\mathbf{\Sigma}$ is impractical to compute or store; however, we might be interested in certain of its elements. For example, the diagonal of $\mathbf{\Sigma}$ contains the variance of each variable and thus, along with the mean, fully describes the posterior marginal densities [29]. Marginal vari-

ances also need to be computed in Gaussian subproblems that arise in the context of non-Gaussian sparse Bayesian learning and relevance vector machine models used for regression, classification, and experimental design [11, 26, 32]. For many of these models variance estimation is the main computational bottleneck in applications involving large scale datasets.

A number of techniques have been proposed for posterior variance estimation. One approach has been to employ modified conjugate gradient algorithms which allow forming variance estimates in parallel to computing the posterior mode when solving the linear system in Eq. (3) [15, 24, 27]. These techniques utilize the close connection between conjugate gradients and the Lanczos method for determining eigensystems [6, 15] but unfortunately exhibit erratic numerical behavior in practice, especially when applied to large scale problems: loss of orthogonality due to finite numerical precision requires that one holds in memory the entire sequence of Lanczos vectors and periodically reorthogonalize them as the iteration progresses, significantly increasing the memory and time complexity relative to ordinary CG; the variance estimates typically converge much slower than mean estimates; one often has limited freedom in initializing the iteration and/or selecting the preconditioner. We refer to [25] for further information.

It is well known that belief propagation computes exact variances in tree-structured GMRFs [36]. However, in graphs with cycles its loopy version typically underestimates the marginal variances since it overcounts the evidence, even when it converges to the correct means [13, 33]. The variance estimator of [28] is only applicable to GMRFs for which just a small number of edges violates the graph's tree structure. The method in [12] relies on a low-rank approximation of the $N \times N$ unit matrix, carefully adapted to the problem covariance structure, also employing a wavelet hierarchy for models exhibiting long-range dependencies. One then needs to solve as many linear systems as is the approximation rank, which in turn increases with the model size ( [12] reports a rank of 448 for a relatively smooth model with about $10^6$ variables). This technique is thus still relatively expensive and not necessarily generally applicable.

The ability to efficiently sample from the Gaussian posterior distribution using the algorithm of Sec. 3 immediately suggests the following Monte Carlo estimator of the posterior covariance matrix

$$\hat{\boldsymbol{\Sigma}} = 1/S \sum\nolimits_{s=1}^{S} (\mathbf{x}_s - \boldsymbol{\mu})(\mathbf{x}_s - \boldsymbol{\mu})^T \,. \tag{5}$$

If only the posterior variances are required, one will obviously just evaluate and retain the diagonal of the outer-products in the sum; any other selected elements of $\hat{\boldsymbol{\Sigma}}$ can similarly be obtained. Clearly, the proposed estimator is unbiased. Its relative variance estimation error follows from the properties of the $\chi^2$ distribution and is $r = \Delta(\hat{\Sigma}_{i,i})/\Sigma_{i,i} = \sqrt{2/S}$. The error drops quite slowly with the number of samples ($S = 2/r^2$ samples are required to reach a desired relative error $r$), so the technique is best suited if rough variance estimates suffice, which is often the case in practical applications [26]; e.g., 50 samples suffice to reduce $r$ to $20\%$. A desirable property of the estimator is that its accuracy is independent of the problem size $N$, in contrast to most alternative techniques. The proposed variance estimation technique can thus be readily applied to every GMRF at a cost of $S$ times that of computing $\boldsymbol{\mu}$. We show in Fig. 2 the result of applying the proposed variance estimator for the thin-membrane GMRF example considered in Sec. 2.3; within only 20 samples (computed in 5 sec.) the qualitative structure of the variance in the model has been captured.

## 5   Block Gibbs sampling in conditionally Gaussian Markov random fields

Following the intuition behind Gaussian sampling by local perturbations, one could try to inject noise to the local potentials and find the mode of the perturbed model, even in the presence of non-quadratic MRF factors. Although such a randomization process is interesting on its own right and deserves further study, it is not feasible to design it in a way that leads to single shot algorithms for exact sampling of non-Gaussian MRFs.

Without completely abandoning the Gaussian realm, we can get versatile models in which some hidden variables $\mathbf{q}$ control the mean and/or variance of the Gaussian factors. Conditional on the values of these hidden variables, the data are still Gaussian

$$P(\mathbf{x}|\mathbf{q}) \propto \prod\nolimits_{l=1}^{L} \mathcal{N}(\mathbf{f}_l^T \mathbf{x}; \mu_{l,\mathbf{q}}, \Sigma_{l,\mathbf{q}}) \,, \tag{6}$$

where we have dropped the dependence on the measurements $\mathbf{y}$ for simplicity. Sampling from this model can be carried out efficiently (but not in a single shot any more) by alternately block sampling from $P(\mathbf{x}|\mathbf{q})$ and $P(\mathbf{q}|\mathbf{x})$, which typically mixes rapidly and is much more efficient than single-site Gibbs sampling [35]. For large models this is feasible because, given the hidden variables, we can update the visible units collectively using the GMRF sampling by local perturbations algorithm, similarly to [16, 23]. We assume that block sampling of the hidden units given the visible variables is also feasible, by considering their conditional distribution independent or tree-structured [16]. One typically employs one discrete hidden variable $q_l$ per factor $\mathbf{f}_l$, leading to mixture of Gaussian local experts for which the joint distribution of visible and hidden units is $P(\mathbf{x}, \mathbf{q}) \propto \prod_{l=1}^{L} \sum_{j=1}^{J_l} \pi_{l,j} \mathcal{N}(\mathbf{f}_l^T \mathbf{x}; \mu_{l,j}, \Sigma_{l,j})$ [4, 16, 23, 34]. Intuitively, the discrete latent unit $q_l$ turns off the smoothness constraint enforced by the factor $\mathbf{f}_l$ by assigning a large variance $\Sigma_{l,j}$ to it when an image edge is detected.

The block-Gibbs sampler leads to a rapidly mixing Markov chain which after a few burn-in iterations generates a sequence of samples $\{\{\mathbf{x}_1, \mathbf{q}_1\}, \ldots, \{\mathbf{x}_S, \mathbf{q}_S\}\}$ that explore the joint distribution $P(\mathbf{x}, \mathbf{q})$. Summarizing the sample sequence into a unique estimate $\hat{\mathbf{x}}$ should be problem dependent. If we strive for minimizing the estimation's mean square error as typically is the case in image denoising, our goal should be to induce the posterior mean from the sample sequence [23]. Apart from the standard sample-based posterior mean estimator $\hat{\mathbf{x}}_S = 1/S \sum_{s=1}^{S} \mathbf{x}_s$, we can alternatively estimate the posterior mean with the Rao-Blackwellized (RB) estimator $\hat{\mathbf{x}}_{RB} = 1/S \sum_{s=1}^{S} E\{\mathbf{x}|\mathbf{q}_s\}$ [16], which offers increased accuracy but requires finding the means of the conditionally Gaussian MRFs $P(\mathbf{x}|\mathbf{q})$, typically doubling the cost per step. Beyond MMSE, in applications such as image inpainting or texture synthesis, the posterior mean can be overly smooth and selecting a single sample from the simulation as the solution can be visually more plausible [8], as can be appreciated by comparing the MMSE and sample reconstructions of the textured areas in the inpainting example of Fig. 2.

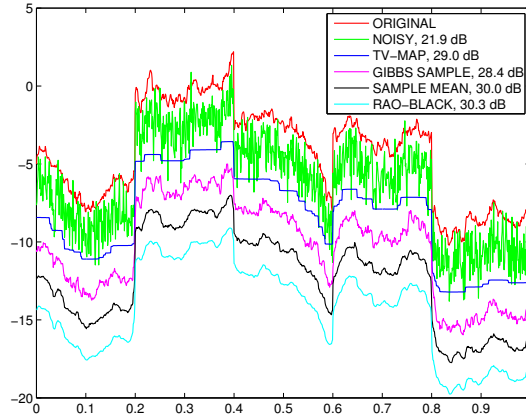

Figure 3: Signal restoration under a total variation prior model and alternative estimation criteria.

The heavy tailed histograms of natural image filter responses are often conveniently approximated by kurtotic continuous parametric distributions [10, 19, 35]. We can still resort to block Gibbs sampling for efficiently exploring the posterior distribution of the signal $\mathbf{x}$ if each expert can be represented as a continuous Gaussian scale mixture (GSM) [2], as has been done before for Student-t experts [35]. Motivated by [14, 23], we show here how this can lead to a novel Bayesian treatment of signal restoration under a total variation (TV) prior $P(\mathbf{x}) \propto \prod_{l=1}^{N-1} \mathcal{L}(\Delta x_l; \alpha)$, which imposes an L1 penalty on the signal differrences $\Delta x_l = x_l - x_{l+1}$. We rely on the hierarchical characterization of the Laplacian density $\mathcal{L}(z; \alpha) = 1/(2\alpha) \exp(-|z|/\alpha)$ as a GSM in which the variance follows an exponential distribution [2, 17]: $\mathcal{L}(z; \alpha) = 1/(2\alpha^2) \int_0^\infty \mathcal{N}(z; 0, v) \exp(-v/\alpha^2) d v$. Thanks to the GSM nature of this representation and assuming a Gaussian measurement model, the conditionally Gaussian visible variables are easy to sample. Further, the latent variances $v_l$ conditionally decouple and have density $P(v_l|\mathbf{x}) \propto v_l^{-1/2} \exp\left(-|\Delta x_l|^2/(2v_l) - v_l/(2\alpha^2)\right)$, which can be recognized as a generalized inverse Gaussian distribution for which standard sampling routines exist. The derivation above carries over to the 2-D TV model, with the gradient magnitude at each pixel replacing $|\Delta x_l|$.

We demonstrate our Bayesian TV restoration method in a signal denoising experiment illustrated in Fig. 3. We synthesized a length-1000 signal by integrating Laplacian noise ($\alpha = 1/8$), also adding jumps of height 5 at four locations (outliers), and subsequently degraded it by adding Gaussian noise (with variance 1). We depict the standard TV-MAP restoration result, as well as plausible solutions extracted from a 10-step block-Gibbs sampling run with our GSM-based Bayesian algorithm: the 10-th sample itself, and the two MMSE estimates outlined above (sample mean and RB). As expected, the two mean estimators are best in terms of PSNR (with the RB one slightly superior). The standard TV-MAP estimator captures the edges more sharply but has lower PSNR score and produces staircase artifacts. Although the random sample performs the worst in terms of PSNR, it resembles most closely the qualitative properties of the original signal, capturing its fine structure. These findings shed new light in the critical view of [14] on MAP-based denoising.

We must emphasize that the block Gibbs sampling strategy outlined above in conjunction with our GMRF sampling by local perturbations algorithm is equally well applicable when the latent variables are distributed, with each hidden variable affecting multiple experts, as illustrated in Fig. 4(a). This situation arises in the context of unsupervised learning of hierarchical models applied on real-valued data, where it is natural to use a Gaussian restricted Boltzmann machine (GRBM) in the first layer of the hierarchy. Training GRBMs with contrastive divergence [7] requires drawing random samples from the model. Sampling the visible layer given the layer of discrete hidden variables is easy if there are no sideways connections between the continuous visible units, as assumed in [9]. To take into account residual correlations among the visible units, the authors of the factored GRBM in [18] drop the conditional independence assumption, but resort to difficult to tune hybrid Monte Carlo (HMC) for sampling. Employing our Gaussian sampling by local perturbations scheme we can efficiently jointly sample the correlated visible units, which allows us to still use the more efficient block-Gibbs sampler in training the model of [18]. To verify this, we have accordingly replaced the sampling module in the publicly available implementation of [18], and have closely followed their setup leaving their model otherwise unchanged. For conditionally Gaussian sampling of the correlated visible units we have used our local perturbation algorithm, coupled with 5 iterations of conjugate gradients running on the GPU. Contrastive divergence training was done on the dataset accompanying their code, which comprises 10240 $16 \times 16$ color patches randomly extracted from the Berkeley dataset and statistically whitened. The receptive fields learned by this procedure are depicted in Fig. 4(b) and look qualitative the same with those reported in [18], while computation time was reduced by a factor of two. Besides this moderate computation gain, the main interest in perturbed Gaussian sampling in this setup lies in its scalability which offers the potential to move beyond the patch-based representation and sample from whole-image factored GRBM models, similarly to what has been recently achieved in [23] for the field of independent experts model [19].

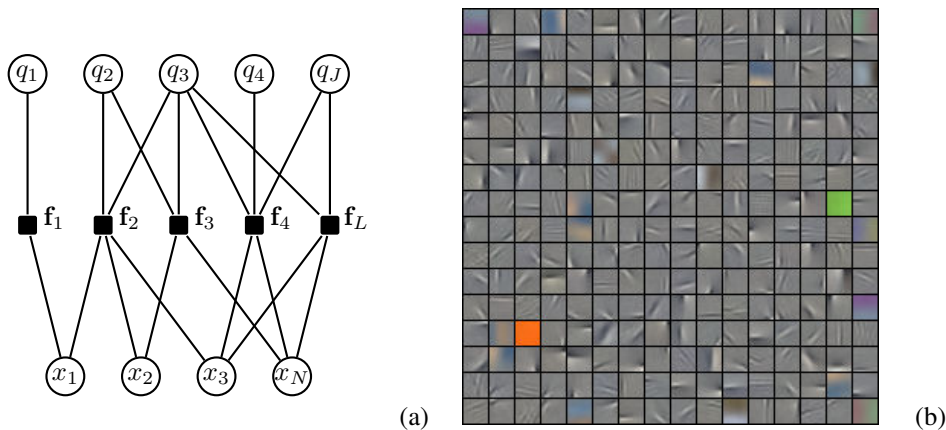

(a)                                                                                       (b)

Figure 4: (a) Each hidden unit can control a single factor (such as the $q_1$ above) or it can affect multiple experts, resulting to models with distributed latent representations. (b) The visible-to-factor filters arising in the factored GRBM model of [18], as learned using block Gibbs sampling.

**Acknowledgments**

This work was supported by the NSF award 0917141 and the AFOSR grant 9550-08-1-0489.

# References

[1] D. Ackley, G. Hinton, and T. Sejnowski. A learning algorithm for Boltzmann machines. *Cogn. Science*, 9(1):147–169, 1985.

[2] D. Andrews and C. Mallows. Scale mixtures of normal distributions. *JRSS (B)*, 36(1):99–102, 1974.

[3] J. Besag. Spatial interaction and the statistical analysis of lattice systems. *JRSS (B)*, 36(2):192–236, 1974.

[4] D. Geman and C. Yang. Nonlinear image recovery with half-quadratic regularization. *IEEE Trans. Image Process.*, 4(7):932–946, 1995.

[5] S. Geman and D. Geman. Stochastic relaxation, Gibbs distributions, and the Bayesian restoration of images. *IEEE Trans. PAMI*, 6(6):721–741, 1984.

[6] G. Golub and C. Van Loan. *Matrix Computations*. John Hopkins Press, 1996.

[7] G. Hinton. Training products of experts by minimizing contrastive divergence. *Neur. Comp.*, 14(8):1771–1800, 2002.

[8] A. Kokaram. *Motion Picture Restoration*. Springer, 1998.

[9] H. Lee, R. Grosse, R. Ranganath, and A. Y. Ng. Convolutional deep belief networks for scalable unsupervised learning of hierarchical representations. In *Proc. ICML*, 2009.

[10] S. Lyu and E. Simoncelli. Modeling multiscale subbands of photographic images with fields of Gaussian scale mixtures. *IEEE Trans. PAMI*, 31(4):693–706, Apr. 2009.

[11] D. MacKay. Bayesian interpolation. *Neur. Comp.*, 4(3):415–447, 1992.

[12] D. Malioutov, J. Johnson, M. Choi, and A. Willsky. Low-rank variance approximation in GMRF models: Single and multiscale approaches. *IEEE Trans. Signal Process.*, 56(10):4621–4634, Oct. 2008.

[13] D. Malioutov, J. Johnson, and A. Willsky. Walk sums and belief propagation in Gaussian graphical models. *J. of Mach. Learning Res.*, 7:2031–2064, 2006.

[14] M. Nikolova. Model distortions in Bayesian MAP reconstruction. *Inv. Pr. and Imag.*, 1(2):399–422, 2007.

[15] C. Paige and M. Saunders. LSQR: An algorithm for sparse linear equations and sparse least squares. *ACM Trans. on Math. Software*, 8(1):43–71, 1982.

[16] G. Papandreou, P. Maragos, and A. Kokaram. Image inpainting with a wavelet domain hidden Markov tree model. In *Proc. ICASSP*, pages 773–776, 2008.

[17] T. Park and G. Casella. The Bayesian lasso. *J. of the Amer. Stat. Assoc.*, 103(482):681–686, 2008.

[18] M. Ranzato, A. Krizhevsky, and G. Hinton. Factored 3-way restricted Boltzmann machines for modeling natural images. In *Proc. AISTATS*, 2010.

[19] S. Roth and M. Black. Fields of experts. *Int. J. of Comp. Vis.*, 82(2):205–229, 2009.

[20] S. Roweis and Z. Ghahramani. A unifying review of linear Gaussian models. *Neur. Comp.*, 11:305–345, 1999.

[21] H. Rue. Fast sampling of Gaussian Markov random fields. *JRSS (B)*, 63(2):325–338, 2001.

[22] H. Rue and L. Held. *Gaussian Markov random fields. Theory and Applications*. Chapman & Hall, 2005.

[23] U. Schmidt, Q. Gao, and S. Roth. A generative perspective on MRFs in low-level vision. In *CVPR*, 2010.

[24] M. Schneider and A. Willsky. Krylov subspace estimation. *SIAM J. Sci. Comp.*, 22(5):1840–1864, 2001.

[25] M. Seeger and H. Nickisch. Large scale variational inference and experimental design for sparse generalized linear models. Technical Report TR-175, MPI for Biological Cybernetics, 2008.

[26] M. Seeger, H. Nickisch, R. Pohmann, and B. Schölkopf. Bayesian experimental design of magnetic resonance imaging sequences. In *NIPS*, pages 1441–1448, 2008.

[27] J. Skilling. Bayesian numerical analysis. In W. Grandy and P. Milonni, editors, *Physics and Probability*, pages 207–221. Cambridge Univ. Press, 1993.

[28] E. Sudderth, M. Wainwright, and A. Willsky. Embedded trees: Estimation of Gaussian processes on graphs with cycles. *IEEE Trans. Signal Process.*, 52(11):3136–3150, Nov. 2004.

[29] R. Szeliski. Bayesian modeling of uncertainty in low-level vision. *Int. J. of Comp. Vis.*, 5(3):271–301, 1990.

[30] R. Szeliski and D. Terzopoulos. From splines to fractals. In *Proc. ACM SIGGRAPH*, pages 51–60, 1989.

[31] D. Terzopoulos. The computation of visible-surface representations. *IEEE Trans. PAMI*, 10(4):417–438, 1988.

[32] M. Tipping. Sparse Bayesian learning and the relevance vector machine. *J. of Mach. Learning Res.*, 1:211–244, 2001.

[33] Y. Weiss and W. Freeman. Correctness of belief propagation in Gaussian graphical models of arbitrary topology. *Neur. Comp.*, 13(10):2173–2200, 2001.

[34] Y. Weiss and W. Freeman. What makes a good model of natural images? In *CVPR*, 2007.

[35] M. Welling, G. Hinton, and S. Osindero. Learning sparse topographic representations with products of Student-t distributions. In *NIPS*, 2002.

[36] A. Willsky. Multiresolution Markov models for signal and image processing. *Proc. IEEE*, 90(8):1396–1458, 2002.

[37] S. Zhu, Y. Wu, and D. Mumford. Filters, random fields and maximum entropy (FRAME): Towards a unified theory for texture modeling. *Int. J. of Comp. Vis.*, 27(2):107–126, 1998.

